# A Hybrid Linear/Nonlinear Approach to Channel Equalization Problems

**Wei-Tsih Lee**    **John Pearson**
David Sarnoff Research Center
CN5300
Princeton, NJ 08543

## Abstract

Channel equalization problem is an important problem in high-speed communications. The sequences of symbols transmitted are distorted by neighboring symbols. Traditionally, the channel equalization problem is considered as a channel-inversion operation. One problem of this approach is that there is no direct correspondence between error probability and residual error produced by the channel inversion operation. In this paper, the optimal equalizer design is formulated as a classification problem. The optimal classifier can be constructed by Bayes decision rule. In general it is nonlinear. An efficient hybrid linear/nonlinear equalizer approach has been proposed to train the equalizer. The error probability of new linear/nonlinear equalizer has been shown to be better than a linear equalizer in an experimental channel.

## 1 INTRODUCTION

In a typical communication system, a sequence of symbols $\{I_i\}$ are transmitted though a linear time-dispersive channel h(t). Let x(t) be the received signal, it can be written as

$$x(t) = \sum_i I_i h(t-nT) + w(t)$$

(1)

where h(t) denotes the elementary pulse waveform, and w(t) represents the random noise with iid Gaussian distribution. In a Quadrature Amplitude Modulation (QAM), symbols $\{I_i\}$ are represented by complex numbers. During the transmission, interferences from neighboring symbols may distort the received signals. It is called Intersymbol Interference (ISI). It mainly because following reasons: nonideal channel which introduces phase or amplitude distortions, phase jitter, and impulse noise. Thus, equalization techniques are used to reduce the ISI.

# 2 ADAPTIVE LINEAR/RADIAL BASIS FUNCTION APPROACH TO EQUALIZER DESIGN

Traditionally, the channel equalization problem is considered as a channel-inversion operation. The idea is that an equalizer is constructed as to undo the interference from neighboring symbols as they passing through a linear dispersive channel. It can be used to explain different equalizer structures (Zero-forcing, Least mean square, and decision feedback) and their performance [Proakis, 1989]. One problem of this approach is that in general there is no direct correspondence between error probability and residual error produced by the channel inversion operation. In [Gibson, et.al, 1991], authors proposed a classification viewpoint for the equalizer design. They suggested that the optimal equalizer should be a classifier whose decision boundary is constructed according to Bayes decision rule. Compared with the channel inversion approach, the outputs of receiver are used as features for a classifier. The decision is made solely based on the classifier output, hence, on feature distribution. As it is well-known in [Fukunaga, 1978], the optimal decision boundaries can rapidly be computed if the features are Gaussian distributed. However, there is no idea about the structure of the optimal equalizer (classifier)for time-dispersive channel outputs. In next section, we prove that for a linear channel, the optimal equalizer is nonlinear.

## 2.1 THE OPTIMAL EQUALIZER OF A LINEAR TIME-DISPERSIVE CHANNEL

Let us first consider a two-value equalization problem. Symbols with two possible values $\{-1, 1\}$ are transmitted. Let the channel be represented in a discrete form as a FIR of $\{h_i\}$, $i=0,N-1$. The output $x_i$ can be written as

$$x_i = \sum_{j=0}^{N-1} I_{i-j}h_j + w_i \tag{2}$$

The optimal equalizer design is equivalent to the following Bayes decision problem. Given $\{x_i\}$, decide $I_i$ by

$$I_i = \left\{ \begin{array}{ll} 1 & if \quad P(I_i = 1| x_i, x_{i+1}, \dots, x_{i+N-1}) \geq P(I_i = -1| x_i, x_{i+1}, \dots, x_{i+N-1}) \\ -1 & if \quad P(I_i = -1| x_i, x_{i+1}, \dots, x_{i+N-1}) > P(I_i = 1| x_i, x_{i+1}, \dots, x_{i+N-1}) \end{array} \right. \tag{3}$$

where $P(I_i = 1| x_i, x_{i+1}, \dots, x_{i+N-1})$ is the posterior probability of the transmitted symbol $I_i$ being 1 given channel output $\{x_i\}$.

By Bayes theorem, expression(3) can be expanded to the following form:

$$P(I_i = 1| x_i, \dots, x_{i+1}x_{i+N-1}) = \frac{P(I_i = 1x_i, x_{i+1}, \dots, x_{i+N-1})}{P(x_i, x_{i+1}, \dots, x_{i+N-1})} \tag{4}$$

$$= \frac{\displaystyle\prod_{j=i}^{i+N-1} \sum_{k_1, k_2, \dots, k_{i-N+1} \in \{1, -1\}} P(x_j| I_i = 1, \dots, I_{i-N+1} = k_{i-N+1})P(I_i = 1, \dots, I_{i-N+1} = k_{i-N+1})}{P(x_i, x_{i+1}, \dots, x_{i+N-1})}$$

$$\tag{5}$$

Since conditional probability $P(x_j| I_i = 1, \dots, I_{i-N+1} = k_{i-N+1})$ in (5) is a Gaussian distribution, the numerator in (5) is a mixture of Gaussian distribution. Plugging (5) into (3), Bayes decision rule determines the optimal decision boundary as the solution of equality. Since denominator is the same on both sides, it can be ignored. Rearranging the equation,

it can be written as summation of exponential functions. The solution of this equation is nonlinear function of $\{x_i\}$. In general, no analytical form can be found. However, it can be solved by numerical methods. Thus, the optimal decision boundary can be determined. The result can be extended to multi-class problems.

Based on the result established above, we provide a theoretical justification of a nonlinear equalizer approach to linear time-dispersive channel. The theoretical comparison of performances of linear and optimal equalizers can be found in [Gibson, et.al, 1991]. They concluded that performance of linear equalizers can not be improved by increasing tap length. This also suggests that a nonlinear equalizer approach is necessary. Another reason for nonlinear equalization approach is due to channels with spectrum hulls [Proakis, 1989]. In this case, the linear equalizer can not achieve the desired performance due to "noise enhancement".

## 2.2 NONLINEAR EQUALIZER DESIGN PROBLEM

There are several approaches to nonlinear equalizer design. To reduce the Least Mean Square (LMS) error, Voterra-series approach uses high-order product terms of input as new features. The tree-structured linear equalizer method [Gelfand, et.al., 1991] partitions the feature-space, and makes a piecewise linear approximation to the optimal nonlinear equalizer. As reported in [Gelfand, et.al., 1991], the tree-structured linear equalizer approach provides reasonable fast convergence and lower error probability as compared with linear and Voterra series approaches. The problem of this approach is that a lot of training samples are needed to achieve good performance. A neural network approach, MultiLayer Perceptron(MLP) [Gibson, et.al, 1991], trains 3 or 4 layers interconnected Perceptrons to form the nonlinear decision boundary. It is observed in [Gibson, et.al, 1991] that the performance of a MLP equalizer is close to optimal Bayes classifier. However, the training time is long and a fine-turing procedure is used. A nonlinear equalizer approach using radial basis functions is also reported in [Chen, et.al., 1991].

To put equalizers into use, the long training time is unpractical, and a fine-adjusting procedure is not allowed. Hence, it is desired to have an efficient, automatic procedure for nonlinear equalize design. To achieve this goal, we propose a hybrid linear and radial basis functions approach for automatic nonlinear equalizer design.

Although the optimal equalizer should be nonlinear, all these nonlinear design methods require long training time or large amount of training samples. Linear equalizers are not optimal, but with following advantages: easy training, fast convergence. It is also reported that the linear equalizer is relatively robust [Fukunaga, 1978]. Hence, it is desirable to combine the advantages of both linear and nonlinear equalizers. However, the hybrid structure should provide desired properties: fast convergence, automatic training procedure, and low error rate. To satisfy these constraints, we propose a feature-space partitioning approach to hybrid equalizer design.

## 2.3 FEATURE-SPACE PARTITIONING APPROACH TO HYBRID EQUALIZER DESIGN

To design a hybrid linear/nonlinear equalizer, we adopt the feature-space partitioning concept. The idea is similar to the one developed in [Gelfand, et.al, 1991]. Here, we consider a partitioning method based on geometrical reasoning for equalization problems. The idea is based on the fact that linear equalizers can recover distorted signals, except the cases when strong noise push samples into boundaries where two classes overlaid with each

other. We consider the "confused" samples as these samples near decision boundaries. The separation of "confused" samples can be accomplished based on the output values of linear equalizers. If the distance between output value and the closest point in signal constellation [Proakis, 1989] is greater than a threshold, then we consider current sample is "confused". This means that the sample is the one close to decision boundary. To achieve an accurate classification, we classify it by a nonlinear equalizer, which is constructed for separating the samples near Bayes decision boundary.

The hybrid structure consisted of a linear equalizer, followed by a radial basis function (RBS) network, as shown in Fig. 1. A RBS network (Fig.2) is a two-layered network with radial_basis_function nodes in first layer, and a weighted linear combination of outputs of these nodes.

Each feature vector consisted of a collection of consecutive data from the channel. It is assumed that these data are properly time and carrier synchronized [Proakis, 1989]. For the QAM, a complex-valued linear equalizer is adopted. The distance between output value of linear equalizer and the closest point is then computed and compared with threshold as described before. The "confused" samples are classified by a nonlinear RBS equalizer. The output of a RBS network can be written as weighted summation of outputs of nodes as follows:

$$f(x) = \sum_i w_i exp \left( -\frac{\|x - c_i\|}{\sigma^2} \right) \qquad (6)$$

where f(x) is the output of network. The output value of each node is computed according to the bell-shaped function centered at $c_i$. $\sigma^2$ is the width of a node. $w_i$ is the weight associated with ith node. In our experiments, the width of all nodes are fixed. The first N training samples are assigned to the centers of N-nodes network. The weights are adjusted according to stochastic gradient decent rule:

$$\Delta w_i = \eta (d_k - f(x_k)) \, exp \left( -\frac{\|x_k - c_i\|}{\sigma^2} \right) \qquad (7)$$

where $\eta$ is the learning rate. $d_k$ is the desired output of network

To train a hybrid LE/RBS equalizer, a collection of training samples is used to adjust the parameters of linear equalizer. The training samples for RBS network are collected according to the distance rule described above. They are used to adjust weights of RBS network only.

The classification of a unknown sample follows a similar rule. The output value of the linear equalizer is computed. If the distance between output value and the closest point in signal constellation is smaller than the threshold, then the closest point is considered as the recovered signal. If not, the output of the RBS network is used to classify the sample. The closest point in signal constellation from output of RBS network is then used for sample class. Note, however, that there is a different interpretation for the output of linear and RBS equalizer. The function of linear equalizers can be considered as an approximation of channel inversion. Hence, it is similar to a deconvolution computation [Proakis, 1989]. However, for a RBS network, the output is the summation of weighted local Gaussian functions. For closely located points, the network is asked to give same output by then training procedure. Thus, it is more a classification approach than a deconvolution method.

The approach provides a design method for hybrid LE/RBS equalizer. The linear equaliz-

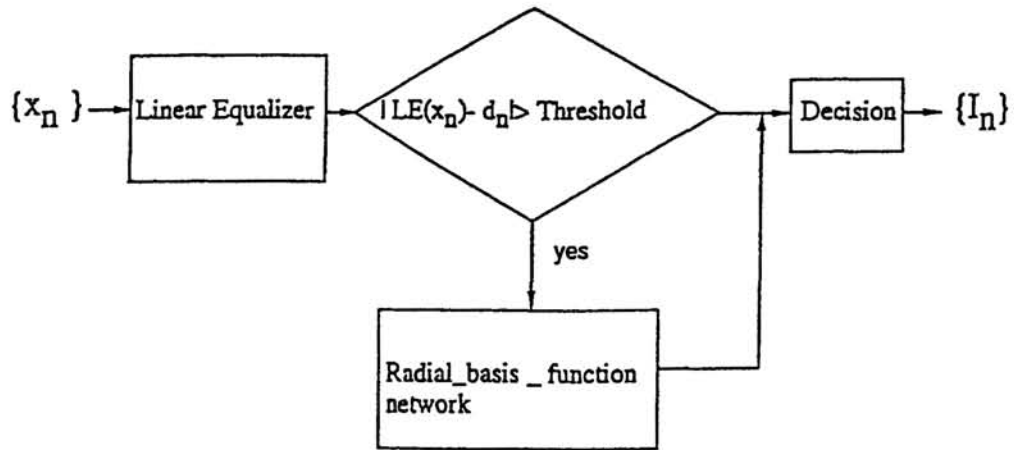

**Fig. 1 System Diagram of Hybrid Linear/Nonlinear Equalizer**

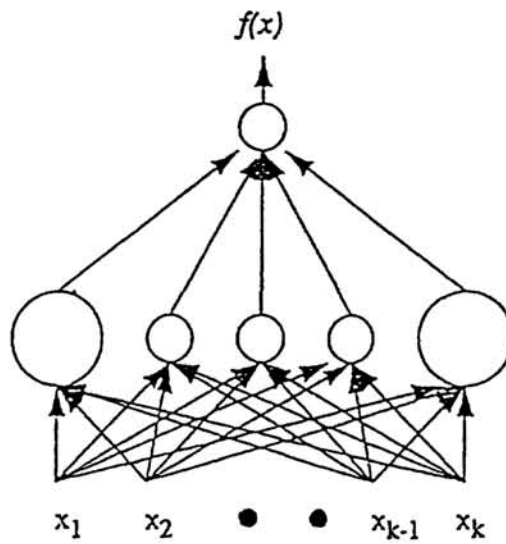

**Fig. 2 A Radial_Basis_Function Network**

ers perform the channel inversion or partitioning of the feature space, depending on the output value. More complicated tree-structured equalizer [Gelfand, et.al., 1991] can be adopted for this proposes. The nonlinear RBS networks are used for classifying "confused" samples. They can be replaced by MLPs. Hence, the approach provides a general method for designing hybrid structure eqalizers. However, the trade-off between complexity and efficiency of these combinations has to to be considered. For example, a multilayer tree-structured equalizer can divide the space into smaller regions for finer classification. However, the small amount of training samples in practice can be a problem for this method. A MLP network can be used for nonlinear classifier. Nevertheless, convergence time will be a major concern.

## 3 EXPERIMENT

We have applied our hybrid design method to a 4-QAM system. The channel is modeled by

$$x_{i-1} = 0.406I_i + 0.814I_{i-1} + 0.407I_{i-2} + w_i \qquad (8)$$

where $I_i = \{-1-j, -1+j, 1-j, 1+j\}$ .

A 7-tapped complex linear equalizer is used for classifying the input. Threshold for nonlinear equalizer is 0.1. We use 4,000training samples and 5,000 testing samples. A 400 nodes RBS network is used for nonlinear equalizer. The first 400 "confused" training samples are used for the center of network. The network is trained according to (6). Learning coefficient $\eta$ is chosen to be 0.01. The width of a RBS node is 1.0.

Fig. 3 shows the symbol error probability vs. SNR. The error probability is evaluated over 5,000 testing samples. The hybrid LE/RBS network produces nearly 10% reduction of error rate compared with linear equalizer. This shows that a hybrid linear/RBS network equalizer can reduce the error rate by classifying "confused" samples near decision boundaries. No comparison with Bayes classifier has been made. In our experiments, it is observed that the error rate can be reduced further by increasing the number of RBS nodes. This seems imply that a large-size RBS network will in general produce better classification result. However, since there is always a limitation of the computation resources: computation time and memory storage, the performance of the hybrid linear/RBS network is limited, especially in high signal constellation case discussed below.

Equalization in high signal constellation, 16 and 64-QAM, have been tried. The result shows no significant improvement. This can be explained by the increasing of complexity. Recall that the RBS network is to separate the samples near the boundary. To deal with the increasing of number of classes due to high signal constellation, the number of nodes of network must increase proportionally. Since the increasing rate is exponential in terms of number of classes, it implies a straight-forward implementation of RBS network method can not be used for high signal constellation. In [Chen, et.al., 1991], authors suggest a dynamical RBS network with adjustable center location and width. The algorithm runs in batch mode. It is reported that the size of network can be reduced dramatically by the dynamical RBS network method. However, for equalizer application, on-line version of the algorithm is needed.

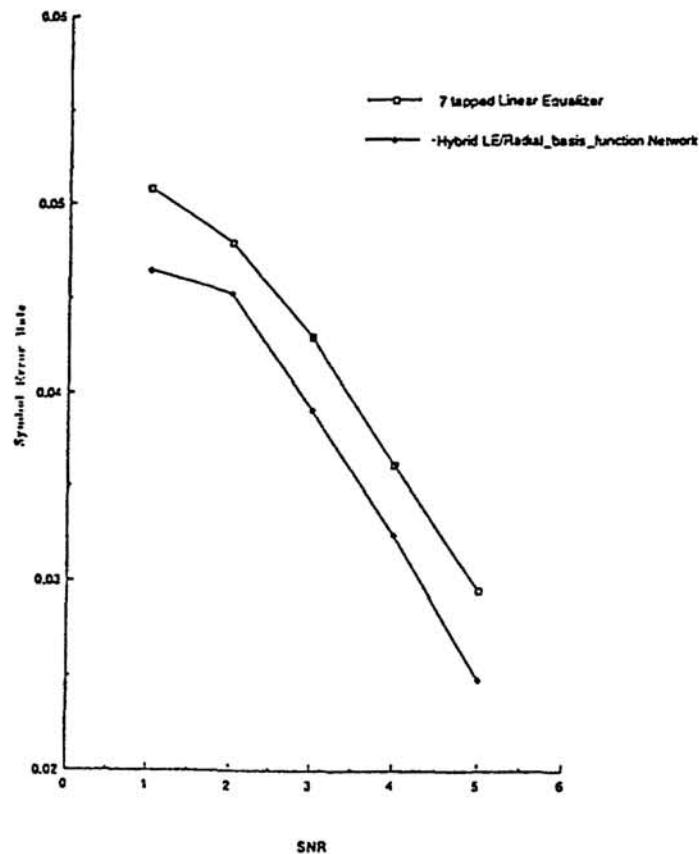

Fig. 3 Error Probability of Hybrid Linear/Radial_Basis_Function Network Equalizer for a Linear Channel $x_{i-1} = 0.406I_i + 0.814I_{i-1} + 0.407I_{i-2} + w_i$ with 4-QAM.

## 4 CONCLUSION AND DISCUSSION FOR HYBRID LE/RBS EQUALIZER DESIGN

By combining feature-space partitioning and nonlinear equalizers, we have developed a hybrid linear/nonlinear equalization approach. The major contribution of this research is to provide a theoretical justification of nonlinear equalization approach for linear time-dispersive channels. A feature-space partitioning method by linear equalizer is proposed. RBS networks for nonlinear equalizers are integrated into the design to separate the samples near decision boundary. The experiments for 4-QAM equalization have demonstrated the feasibility of the approach. For high signal constellation modulation, a dynamical RBS network method [Chen, et.al., 1991] has been suggested to overcome the problem of increasing complexity.

The hybrid Linear/nonlinear equalization approach combines the strength of linear and nonlinear equalizers. It offers a framework to integrate the deconvolution and classification methods. The approach can be generalized to include complicated partitioning

scheme and other nonlinear networks, such as MLP, as well.

More researches need to be conducted to make this approach practical for general use. The relationship between the performance of hybrid equalizer and taps length of linear equalizers, the width and the number of RBS nodes need to be investigated. The on-line version of dynamical RBS network [Chen, et.al., 1991] need to be developed.

## Reference:

Proakis, J. G., *Digital Communications*, McGrwa-Hill company, New York, 1989.

Gibson, G.J., Siu, S., Cowan, C.F.N., "The Application of Nonlinear Structures to Reconstruction of Binary Signals," IEEE. Trans. on Signal Processing, vol. 39, No. 8, Aug.,pp. 1877-1884, 1991.

Fukunaga, K., Introduction to Statistical Pattern Recognition, Academic Press, New York, 1978.

Gelfand, S.B., Ravishankar, C.S., and Delp, E.J., "Tree-structured Piecewise Linear Adaptive Equalization," ICC91, 001383, 1386.

Chen, S., Gilbson, G.J., Cowan, C.F.N., and Grant, P.M., "Reconstruction of binary signals using an adaptive radial-basis-function equalizer," Signal Processing, 22, pp. 77-93, 1991.

Chen, S., Cowan, C.F.N., and Grant, P.M., "Orthogonal Least Squares Learning Algorithm for Radial Basis Function Networks," IEEE. Trans. on Neural Networks, vol. 2, no., 2, March, pp.302-309, 1991.